# Demixed Principal Component Analysis

**Wieland Brendel**
Ecole Normale Supérieure, Paris, France
Champalimaud Neuroscience Programme
Lisbon, Portugal

**Ranulfo Romo**
Instituto de Fisiología Celular
Universidad Nacional Autónoma de México
Mexico City, Mexico

**Christian K. Machens**
Ecole Normale Supérieure, Paris, France
Champalimaud Neuroscience Programme, Lisbon, Portugal

## Abstract

In many experiments, the data points collected live in high-dimensional observation spaces, yet can be assigned a set of labels or parameters. In electrophysiological recordings, for instance, the responses of populations of neurons generally depend on mixtures of experimentally controlled parameters. The heterogeneity and diversity of these parameter dependencies can make visualization and interpretation of such data extremely difficult. Standard dimensionality reduction techniques such as principal component analysis (PCA) can provide a succinct and complete description of the data, but the description is constructed independent of the relevant task variables and is often hard to interpret. Here, we start with the assumption that a particularly informative description is one that reveals the dependency of the high-dimensional data on the individual parameters. We show how to modify the loss function of PCA so that the principal components seek to capture both the maximum amount of variance about the data, while also depending on a minimum number of parameters. We call this method demixed principal component analysis (dPCA) as the principal components here segregate the parameter dependencies. We phrase the problem as a probabilistic graphical model, and present a fast Expectation-Maximization (EM) algorithm. We demonstrate the use of this algorithm for electrophysiological data and show that it serves to demix the parameter-dependence of a neural population response.

## 1   Introduction

Samples of multivariate data are often connected with labels or parameters. In fMRI data or electrophysiological data from awake behaving humans and animals, for instance, the multivariate data may be the voxels of brain activity or the firing rates of a population of neurons, and the parameters may be sensory stimuli, behavioral choices, or simply the passage of time. In these cases, it is often of interest to examine how the external parameters or labels are represented in the data set.

Such data sets can be analyzed with principal component analysis (PCA) and related dimensionality reduction methods [4, 2]. While these methods are usually successful in reducing the dimensionality of the data, they do not take the parameters or labels into account. Not surprisingly, then, they often fail to represent the data in a way that simplifies the interpretation in terms of the underlying parameters. On the other hand, dimensionality reduction methods that can take parameters into account, such as canonical correlation analysis (CCA) or partial least squares (PLS) [1, 5], impose a specific model of how the data depend on the parameters (e.g. linearly), which can be too restrictive.

We illustrate these issues with neural recordings collected from the prefrontal cortex (PFC) of monkeys performing a two-frequency discrimination task [9, 3, 7]. In this experiment a monkey received

two mechanical vibrations with frequencies $f_1$ and $f_2$ on its fingertip, delayed by three seconds. The monkey then had to make a binary decision $d$ depending on whether $f_1 > f_2$. In the data set, each neuron has a unique firing pattern, leading to a large diversity of neural responses. The firing rates of three neurons (out of a total of 842) are plotted in Fig. 1, top row. The responses of the neurons mix information about the different task parameters, a common observation for data sets of recordings in higher-order brain areas, and a problem that exacerbates interpretation of the data.

Here we address this problem by modifying PCA such that the principal components depend on individual task parameters while still capturing as much variance as possible. Previous work has addressed the question of how to demix data depending on two [7] or several parameters [8], but did not allow components that capture nonlinear mixtures of parameters. Here we extend this previous work threefold: (1) we show how to systematically split the data into univariate and multivariate parameter dependencies; (2) we show how this split suggests a simple loss function, capable of demixing data with arbitrary combinations of parameters, (3) we introduce a probabilistic model for our method and derive a fast algorithm using expectation-maximization.

## 2 Principal component analysis and the demixing problem

The firing rates of the neurons in our dataset depend on three external parameters: the time $t$, the stimulus $s = f_1$, and the decision $d$ of the monkey. We omit the second frequency $f_2$ since this parameter is highly correlated with $f_1$ and $d$ (the monkey makes errors in $< 10\%$ of the trials). Each sample of firing rates in the population, $\mathbf{y}_n$, is therefore tagged with parameter values $(t_n, s_n, d_n)$. For notational simplicity, we will assume that each data point is associated with a unique set of parameter values so that the parameter values themselves can serve as indices for the data points $\mathbf{y}_n$. In turn, we drop the index $n$, and simply write $\mathbf{y}_{tsd}$.

The main aim of PCA is to find a new coordinate system in which the data can be represented in a more succinct and compact fashion. The covariance matrix of the firing rates summarizes the second-order statistics of the data set,

$$\mathbf{C} = \left\langle \mathbf{y}_{tsd} \mathbf{y}_{tsd}^\top \right\rangle_{tsd} \tag{1}$$

and has size $D \times D$ where $D$ is the number of neurons in the data set (we will assume the data are centered throughout the paper). The angular bracket denotes averaging over all parameter values $(t, s, d)$ which corresponds to averaging over all data points. Given the covariance matrix, we can compute the firing rate variance that falls along arbitrary directions in state space. For instance, the variance captured by a coordinate axis given by a normalized vector $\mathbf{w}$ is simply

$$L = \mathbf{w}^\top \mathbf{C} \mathbf{w}. \tag{2}$$

The first principal component corresponds to the axis that captures most of the variance of the data, and thereby maximizes the function $L$ subject to the normalization constraint $\mathbf{w}^\top \mathbf{w} = 1$. The second principal component maximizes variance in the orthogonal subspace and so on [4, 2].

PCA succeeds nicely in summarizing the population response for our data set: the first ten principal components capture more than 90% of the variance of the data. However, PCA completely ignores the causes of firing rate variability. Whether firing rates have changed due to the first stimulus frequency $s = f_1$, due to the passage of time, $t$, or due to the decision, $d$, they will enter equally into the computation of the covariance matrix and therefore do not influence the choice of the coordinate system constructed by PCA. To clarify this observation, we will segregate the data $\mathbf{y}_{tsd}$ into pieces capturing the variability caused by different parameters.

**Marginalized average.**  Let us denote the set of parameters by $S = \{t, s, d\}$. For every subset of $S$ we construct a 'marginalized average',

$$\bar{\mathbf{y}}_t := \langle \mathbf{y}_{tsd} \rangle_{sd}, \qquad \bar{\mathbf{y}}_s := \langle \mathbf{y}_{tsd} \rangle_{td}, \qquad \bar{\mathbf{y}}_d := \langle \mathbf{y}_{tsd} \rangle_{ts} \tag{3}$$

$$\bar{\mathbf{y}}_{ts} := \langle \mathbf{y}_{tsd} \rangle_d - \bar{\mathbf{y}}_t - \bar{\mathbf{y}}_s, \qquad \bar{\mathbf{y}}_{td} := \langle \mathbf{y}_{tsd} \rangle_s - \bar{\mathbf{y}}_t - \bar{\mathbf{y}}_d, \qquad \bar{\mathbf{y}}_{sd} := \langle \mathbf{y}_{tsd} \rangle_t - \bar{\mathbf{y}}_s - \bar{\mathbf{y}}_d, \tag{4}$$

$$\bar{\mathbf{y}}_{tsd} := \mathbf{y}_{tsd} - \bar{\mathbf{y}}_{ts} - \bar{\mathbf{y}}_{td} - \bar{\mathbf{y}}_{sd} - \bar{\mathbf{y}}_t - \bar{\mathbf{y}}_s - \bar{\mathbf{y}}_d, \tag{5}$$

where $\langle \mathbf{y}_{tsd} \rangle_\phi$ denotes the average of the data over the subset $\phi \subseteq S$. In $\bar{\mathbf{y}}_t = \langle \mathbf{y}_{tsd} \rangle_{sd}$, for instance, we average over all parameter values $(s, d)$ such that the remaining variation of the averaged data

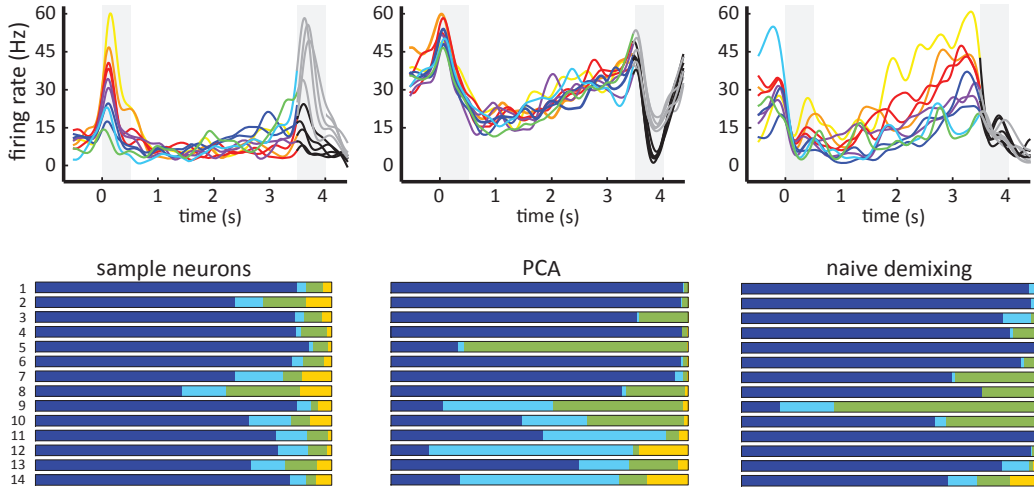

Figure 1: (Top row) Firing rates of three (out of $D = 842$) neurons recorded in the PFC of monkeys discriminating two vibratory frequencies. The two stimuli were presented during the shaded periods. The rainbow colors indicate different stimulus frequencies $f_1$, black and gray indicate the decision of the monkey during the interval [3.5,4.5] sec. (Bottom row) Relative contribution of time (blue), stimulus (light blue), decision (green), and non-linear mixtures (yellow) to the total variance for a sample of 14 neurons (left), the top 14 principal components (middle), and naive demixing (right).

only comes from $t$. In $\bar{\mathbf{y}}_{ts}$, we subtract all variation due to $t$ or $s$ individually, leaving only variation that depends on combined changes of $(t, s)$. These marginalized averages are orthogonal so that

$$\forall \phi, \phi' \subseteq S \quad \langle \bar{\mathbf{y}}_\phi^\top \bar{\mathbf{y}}_{\phi'} \rangle = 0 \quad \text{if} \quad \phi \neq \phi'. \tag{6}$$

At the same time, their sum reconstructs the original data,

$$\mathbf{y}_{tsd} = \bar{\mathbf{y}}_t + \bar{\mathbf{y}}_s + \bar{\mathbf{y}}_d + \bar{\mathbf{y}}_{ts} + \bar{\mathbf{y}}_{td} + \bar{\mathbf{y}}_{sd} + \bar{\mathbf{y}}_{tsd}. \tag{7}$$

The latter two properties allow us to segregate the covariance matrix of $\mathbf{y}_{tsd}$ into 'marginalized covariance matrices' that capture the variance in a subset of parameters $\phi \subseteq S$,

$$\mathbf{C} = \mathbf{C}_t + \mathbf{C}_s + \mathbf{C}_d + \mathbf{C}_{ts} + \mathbf{C}_{td} + \mathbf{C}_{sd} + \mathbf{C}_{tsd}, \quad \text{with} \quad \mathbf{C}_\phi = \langle \bar{\mathbf{y}}_\phi \bar{\mathbf{y}}_\phi^\top \rangle.$$

Note that here we use the parameters $\{t, s, d\}$ as labels, whereas they are indices in Eq. (3)-(5), and (7). For a given component $\mathbf{w}$, the marginalized covariance matrices allow us to calculate the variance $x_\phi$ of $\mathbf{w}$ conditioned on $\phi \subseteq S$ as

$$x_\phi^2 = \mathbf{w}^\top \mathbf{C}_\phi \mathbf{w},$$

so that the total variance is given by $L = \sum_\phi x_\phi^2 =: \|\mathbf{x}\|_2^2$.

Using this segregation, we are able to examine the distribution of variance in the PCA components and the original data. In Fig. 1, bottom row, we plot the relative contributions of time (blue; computed as $x_t^2 / \|\mathbf{x}\|_2^2$), decision (light blue; computed as $(x_d^2 + x_{td}^2)/\|\mathbf{x}\|_2^2$), stimulus (green; computed as $(x_s^2 + x_{ts}^2)/\|\mathbf{x}\|_2^2$), and nonlinear mixtures of stimulus and decision (yellow; computed as $(x_{sd}^2 + x_{tsd}^2)/\|\mathbf{x}\|_2^2$) for a set of sample neurons (left) and for the first fourteen components of PCA (center). The left plot shows that individual neurons carry varying degree of information about the different task parameters, reaffirming the heterogeneity of neural responses. While the situation is slightly better for the PCA components, we still find a strong mixing of the task parameters.

To improve visualization of the data and to facilitate the interpretation of individual components, we would prefer components that depend on only a single parameter, or, more generally, that depend on the smallest number of parameters possible. At the same time, we would want to keep the attractive properties of PCA in which every component captures as much variance as possible about the data.

Naively, we could simply combine eigenvectors from the marginalized covariance matrices. For example, consider the first $Q$ eigenvectors of each marginalized covariance matrix. Apply symmetric

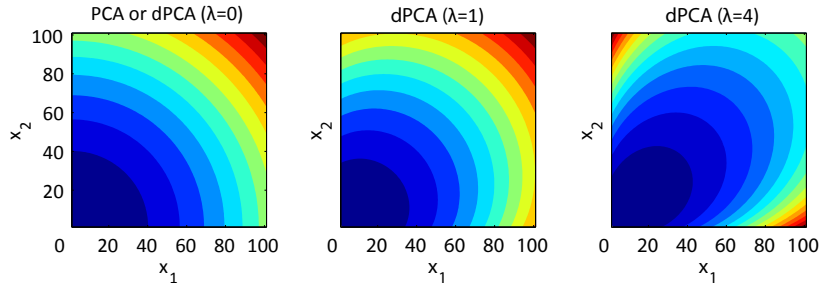

Figure 2: Illustration of the objective functions. The PCA objective function corresponds to the L2-norm in the space of standard deviations, $\mathbf{x}$. Whether a solution falls into the center or along the axis does not matter, as long as it captures a maximum of overall variance. The dPCA objective functions (with parameters $\lambda = 1$ and $\lambda = 4$) prefer solutions along the axes over solutions in the center, even if the solutions along the axes capture less overall variance.

orthogonalization to these eigenvectors and choose the $Q$ coordinates that capture the most variance. The resulting variance distribution is plotted in Fig. 1 (bottom, right). While the parameter dependence of the components is sparser than in PCA, there is a strong bias towards time, and variance induced by the decision of the monkey is squeezed out. As a further drawback, naive demixing covers only 84.6% of the total variance compared with 91.7% for PCA. We conclude that we have to rely on a more systematic approach based specifically on an objective that promotes demixing.

## 3  Demixed principal component analysis (dPCA): Loss function

With respect to the segregated covariances, the PCA objective function, Eq. (2), can be written as $L = \mathbf{w}^\top C \mathbf{w} = \sum_\phi \mathbf{w}^\top C_\phi \mathbf{w} = \sum_\phi x_\phi^2 = \|\mathbf{x}\|_2^2$. This function is illustrated in Fig 2 (left), and shows that PCA will maximize variance, no matter whether this variance comes about through a single marginalized variance, or through mixtures thereof.

Consequently, we need to modify this objective function such that solutions $\mathbf{w}$ that do not mix variances—thereby falling along one of the axes in $\mathbf{x}$-space—are favored over solutions $\mathbf{w}$ that fall into the center in $\mathbf{x}$-space. Hence, we seek an objective function $L = L(\mathbf{x})$ that grows monotonically with any $x_\phi$ such that more variance is better, just as in PCA, and that grows faster along the axes than in the center so that mixtures of variances get punished. A simple way of imposing this is

$$L_{\text{dPCA}} = \|\mathbf{x}\|_2^2 \left( \frac{\|\mathbf{x}\|_2}{\|\mathbf{x}\|_1} \right)^\lambda \tag{8}$$

where $\lambda \geq 0$ controls the tradeoff. This objective function is illustrated in Fig. 2 (center and right) for two values of $\lambda$. Here, solutions $\mathbf{w}$ that lead to mixtures of variances are punished against solutions that do not mix variances.

Note that the objective function is a function of the coordinate axis $\mathbf{w}$, and the aim is to maximize $L_{\text{dPCA}}$ with respect to $\mathbf{w}$. A generalization to a set of $Q$ components $\mathbf{w}_1, \ldots, \mathbf{w}_Q$ is straightforward by maximizing $L$ in steps for every component and ensuring orthonormality by means of symmetric orthogonalization [6] after each step. We call the resulting algorithm *demixed* principal component analysis (dPCA), since it essentially can be seen as a generalization of standard PCA.

## 4  Probabilistic principal component analysis with orthogonality constraint

We introduced dPCA by means of a modification of the objective function of PCA. It is straightforward to build a gradient ascent algorithm to solve Eq. (8). However, we aim for a superior algorithm by framing dPCA in a probabilistic framework. A probabilistic model provides several benefits that include dealing with missing data and the inclusion of prior knowledge [see 2, p. 570]. Since the probabilistic treatment of dPCA requires two modifications over the conventional expectation-maximization (EM) algorithm for probabilistic PCA (PPCA), we here review PPCA [11, 10], and show how to introduce an explicit orthogonality constraint on the mixing matrix.

In PPCA, the observed data $\mathbf{y}$ are linear combinations of latent variables $\mathbf{z}$

$$\mathbf{y} = \mathbf{W}\mathbf{z} + \boldsymbol{\epsilon}_y \qquad (9)$$

where $\boldsymbol{\epsilon}_y \sim \mathcal{N}(\mathbf{0}, \sigma^2 \mathrm{I}_D)$ is isotropic Gaussian noise with variance $\sigma^2$ and $\mathbf{W} \in \mathbb{R}^{D \times Q}$ is the mixing matrix. In turn, $p(\mathbf{y}|\mathbf{z}) = \mathcal{N}(\mathbf{y}|\mathbf{W}\mathbf{z}, \sigma^2 \mathrm{I}_D)$. The latent variables are assumed to follow a zero-mean, unit-covariance Gaussian prior, $p(\mathbf{z}) = \mathcal{N}(\mathbf{z}|\mathbf{0}, \mathrm{I}_Q)$. These equations completely specify the model of the data and allow us to compute the marginal distribution $p(\mathbf{y})$.

Let $\mathrm{Y} = \{\mathbf{y}_n\}$ be the set of data points, with $n = 1 \dots N$, and $\mathrm{Z} = \{\mathbf{z}_n\}$ the corresponding values of the latent variables. Our aim is to maximize the likelihood of the data, $p(\mathrm{Y}) = \prod_n p(\mathbf{y}_n)$, with respect to the parameters $\mathbf{W}$ and $\sigma$. To this end, we use the EM algorithm, in which we first calculate the statistics (mean and covariance) of the posterior distribution, $p(\mathrm{Z}|\mathrm{Y})$, given fixed values for $\mathbf{W}$ and $\sigma^2$ (Expectation step). Then, using these statistics, we compute the expected complete-data likelihood, $\mathbb{E}[p(\mathrm{Y}, \mathrm{Z})]$, and maximize it with respect to $\mathbf{W}$ and $\sigma^2$ (Maximization step). We cycle through the two steps until convergence.

**Expectation step.** The posterior distribution $p(\mathrm{Z}|\mathrm{Y})$ is again Gaussian and given by

$$p(\mathrm{Z}|\mathrm{Y}) = \prod_{n=1}^{N} \mathcal{N}\left(\mathbf{z}_n \big| \mathrm{M}^{-1}\mathbf{W}^{\top}\mathbf{y}_n, \sigma^2 \mathrm{M}^{-1}\right) \quad \text{with} \quad \mathrm{M} = \mathbf{W}^{\top}\mathbf{W} + \sigma^2 \mathrm{I}_Q. \qquad (10)$$

Mean and covariance can be read off the arguments, and we note in particular that $\mathbb{E}[\mathbf{z}_n \mathbf{z}_n^{\top}] = \sigma^2 \mathrm{M}^{-1} + \mathbb{E}[\mathbf{z}_n]\mathbb{E}[\mathbf{z}_n]^{\top}$. We can then take the expectation of the complete-data log likelihood with respect to this posterior distribution, so that

$$\mathbb{E}\left[\ln p\left(\mathrm{Y}, \mathrm{Z}|\mathbf{W}, \sigma^2\right)\right] = -\sum_{n=1}^{N} \left\{ \frac{D}{2} \ln\left(2\pi\sigma^2\right) + \frac{1}{2\sigma^2}\|\mathbf{y}_n\|^2 - \frac{1}{\sigma^2}\mathbb{E}\left[\mathbf{z}_n\right]^{\top}\mathbf{W}^{\top}\mathbf{y}_n \right.$$
$$\left. + \frac{1}{2\sigma^2}\mathrm{Tr}\left(\mathbb{E}\left[\mathbf{z}_n\mathbf{z}_n^{\top}\right]\mathbf{W}^{\top}\mathbf{W}\right) + \frac{Q}{2}\ln\left(2\pi\right) + \frac{1}{2}\mathrm{Tr}\left(\mathbb{E}\left[\mathbf{z}_n\mathbf{z}_n^{\top}\right]\right) \right\}. \qquad (11)$$

**Maximization step.** Next, we need to maximize Eq. (11) with respect to $\sigma$ and $\mathbf{W}$. For $\sigma$, we obtain

$$(\sigma^*)^2 = \frac{1}{ND}\sum_{n=1}^{N}\left\{\|\mathbf{y}_n\|^2 - 2\mathbb{E}\left[\mathbf{z}_n\right]^{\top}\mathbf{W}^{\top}\mathbf{y}_n + \mathrm{Tr}\left(\mathbb{E}\left[\mathbf{z}_n\mathbf{z}_n^{\top}\right]\mathbf{W}^{\top}\mathbf{W}\right)\right\}. \qquad (12)$$

For $\mathbf{W}$, we need to deviate from the conventional PPCA algorithm, since the development of probabilistic dPCA requires an explicit orthogonality constraint on $\mathbf{W}$, which had so far not been included in PPCA. To impose this constraint, we factorize $\mathbf{W}$ into an orthogonal and a diagonal matrix,

$$\mathbf{W} = \mathrm{U}\Gamma, \quad \mathrm{U}^{\top}\mathrm{U} = \mathrm{I}_D \qquad (13)$$

where $\mathrm{U} \in \mathbb{R}^{D \times Q}$ has orthogonal columns of unit length and $\Gamma \in \mathbb{R}^{Q \times Q}$ is diagonal. In order to maximize Eq. (11) with respect to $\mathrm{U}$ and $\Gamma$ we make use of infinitesimal translations in the respective restricted space of matrices,

$$\mathrm{U} \to (\mathrm{I}_D + \epsilon \mathrm{A})\,\mathrm{U}, \qquad \Gamma \to (\mathrm{I}_Q + \epsilon\,\mathrm{diag}(\mathbf{b}))\,\Gamma, \qquad (14)$$

where $\mathrm{A} \in \mathrm{Skew}_D$ is $D \times D$ skew-symmetric, $\mathbf{b} \in \mathbb{R}^Q$, and $\epsilon \ll 1$. The set of $D \times D$ skew-symmetric matrices are the generators of rotations in the space of orthogonal matrices. The necessary conditions for a maximum of the likelihood function at $U^*, \Gamma^*$ are

$$\mathbb{E}\left[\ln p\left(\mathrm{Y}, \mathrm{Z}\big|(\mathrm{I}_D + \epsilon\mathrm{A})\,\mathrm{U}^*\Gamma, \sigma^2\right)\right] - \mathbb{E}\left[\ln p\left(\mathrm{Y}, \mathrm{Z}|\mathrm{U}^*\Gamma, \sigma^2\right)\right] = 0 + \mathcal{O}\left(\epsilon^2\right) \;\; \forall\mathrm{A} \in \mathrm{Skew}_D, \qquad (15)$$

$$\mathbb{E}\left[\ln p\left(\mathrm{Y}, \mathrm{Z}|\mathrm{U}\left(\mathrm{I}_Q + \epsilon\,\mathrm{diag}(\mathbf{b})\right)\Gamma^*, \sigma^2\right)\right] - \mathbb{E}\left[\ln p\left(\mathrm{Y}, \mathrm{Z}|\mathrm{U}\Gamma^*, \sigma^2\right)\right] = 0 + \mathcal{O}\left(\epsilon^2\right) \;\; \forall\mathbf{b} \in \mathbb{R}^D. \qquad (16)$$

Given the *reduced* singular value decomposition[1] of $\sum_n \mathbf{y}_n \mathbb{E}\left[\mathbf{z}_n^{\top}\right]\Gamma = \mathrm{K}\Sigma\mathrm{L}^{\top}$, the maximum is

$$\mathrm{U}^* = \mathrm{K}\mathrm{L}^{\top} \qquad (17)$$

$$\Gamma^* = \mathrm{diag}\left(\mathrm{U}^{\top}\sum_n \mathbf{y}_n \mathbb{E}\left[\mathbf{z}_n^{\top}\right]\right)\mathrm{diag}\left(\sum_n \mathbb{E}\left[\mathbf{z}_n\mathbf{z}_n^{\top}\right]\right)^{-1} \qquad (18)$$

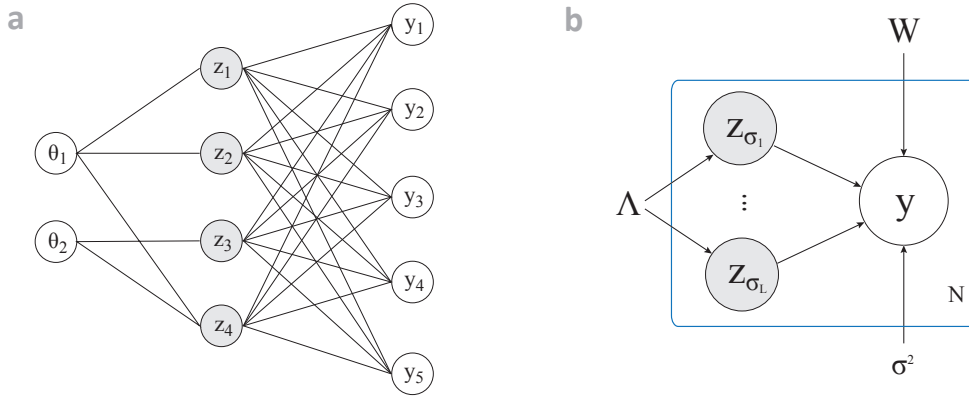

Figure 3: (a) Graphical representation of the general idea of dPCA. Here, the data $\mathbf{y}$ are projected on a subspace $\mathbf{z}$ of latent variables. Each latent variable $z_i$ depends on a set of parameters $\theta_j \in S$. To ease interpretation of the latent variables $z_i$, we impose a sparse mapping between the parameters and the latent variables. (b) Full graphical model of dPCA.

where diag(A) returns a square matrix with the same diagonal as A but with all off-diagonal elements set to zero.

## 5  Probabilistic demixed principal component analysis

We described a PPCA EM-algorithm with an explicit constraint on the orthogonality of the columns of W. So far, variance due to different parameters in the data set are completely mixed in the latent variables $\mathbf{z}$. The essential idea of dPCA is to demix these parameter dependencies by sparsifying the mapping from parameters to latent variables (see Fig. 3a). Since we do not want to impose the nature of this mapping (which is to remain non-parametric), we suggest a model in which each latent variable $z_i$ is segregated into (and replaced by) a set of $R$ latent variables $\{z_{\phi,i}\}$, each of which depends on a subset $\phi \subseteq S$ of parameters. Note that $R$ is the number of all subsets of $S$, exempting the empty set. We require $z_i = \sum_{\phi \subseteq S} z_{\phi,i}$, so that

$$\mathbf{y} = \sum_{\phi \subseteq S} \mathbf{W}\mathbf{z}_\phi + \boldsymbol{\epsilon}_y \tag{19}$$

with $\boldsymbol{\epsilon}_y \sim \mathcal{N}(\mathbf{0}, \sigma^2 I_D)$, see also Fig. 3b. The priors over the latent variables are specified as

$$p(\mathbf{z}_\phi) = \mathcal{N}\left(\mathbf{z}_\phi | \mathbf{0}, \mathrm{diag}\boldsymbol{\Lambda}_\phi\right) \tag{20}$$

where $\boldsymbol{\Lambda}_\phi$ is a row in $\Lambda \in \mathbb{R}^{R \times Q}$, the matrix of variances for all latent variables. The covariance of the sum of the latent variables shall again be the identity matrix,

$$\sum_{\phi \subseteq S} \mathrm{diag}\, \boldsymbol{\Lambda}_\phi = I_Q. \tag{21}$$

This completely specifies our model. As before, we will use the EM-algorithm to maximize the model evidence $p(\mathbf{Y})$ with respect to the parameters $\Lambda, \mathrm{W}, \sigma$. However, we additionally impose that each column $\boldsymbol{\Lambda}_i$ of $\Lambda$ shall be sparse, thereby ensuring that the diversity of parameter dependencies of the latent variables $z_i = \sum_\phi z_{\phi,i}$ is reduced. Note that $\boldsymbol{\Lambda}_i$ is proportional to the vector $\mathbf{x}$ with elements $x_\phi$ introduced in section 3. This links the probabilistic model to the loss function in Eq. (8).

**Expectation step**. Due to the implicit parameter dependencies of the latent variables, the sets of variables $\mathrm{Z}_\phi = \{\mathbf{z}_\phi^n\}$ can only depend on the respective marginalized averages of the data. The posterior distribution over all latent variables $\mathrm{Z} = \{\mathrm{Z}_\phi\}$ therefore factorizes such that

$$p(\mathrm{Z}|\mathrm{Y}) = \prod_{\phi \subseteq S} p(\mathrm{Z}_\phi | \bar{\mathrm{Y}}_\phi) \tag{22}$$

---

**Algorithm 1:** demixed Principal Component Analysis (dPCA)

---

**Input:** Data Y, # components $Q$

**Algorithm:**

    $\mathrm{U}^{(k=1)} \leftarrow$ first $Q$ principal components of $\mathbf{y}$,   $\mathrm{I}^{(k=1)} \leftarrow \mathrm{I}_Q$

**repeat**

    $\mathrm{M}_\phi^{(k)}, \mathrm{U}^{(k)}, \Gamma^{(k)}, \sigma^{(k)}, \Lambda^{(k)} \rightarrow$ update using (25), (17), (18), (12) and (30)

    $k \leftarrow k + 1$

**until** $p(\mathrm{Y})$ converges

---

where $\bar{\mathrm{Y}}_\phi = \{\bar{\mathbf{y}}_\phi^n\}$ are the marginalized averages over the complete data set. For three parameters, the marginalized averages were specified in Eq. (3)-(7). For more than three parameters, we obtain

$$\bar{\mathbf{y}}_\phi^n = \langle\mathbf{y}\rangle_{(S\setminus\phi)}^n + \sum_{\tau \subseteq \phi} (-1)^{|\tau|} \langle\mathbf{y}\rangle_{(S\setminus\phi)\cup\tau}^n . \tag{23}$$

where $\langle\mathbf{y}\rangle_\psi^n$ denotes averaging of the data over the parameter subset $\psi$. The index $n$ refers the average to the respective data point.[2] In turn, the posterior of $Z_\phi$ takes the form

$$p(\mathrm{Z}_\phi|\bar{\mathrm{Y}}_\phi) = \prod_{n=1}^N \mathcal{N}\left(\mathbf{z}_\phi^n \middle| \mathrm{M}_\phi^{-1}\mathrm{W}^\top\bar{\mathbf{y}}_\phi^n, \sigma^2\mathrm{M}_\phi^{-1}\right) \tag{24}$$

where

$$\mathrm{M}_\phi = \mathrm{W}^\top\mathrm{W} + \sigma^2 \,\mathrm{diag}\,\boldsymbol{\Lambda}_\phi^{-1}. \tag{25}$$

Hence, the expectation of the complete-data log-likelihood function is modified from Eq. (11),

$$\begin{aligned}
\mathbb{E}\left[\ln p\left(\mathrm{Y}, \mathrm{Z}\middle|\mathrm{W}, \sigma^2\right)\right] = -\sum_{n=1}^N \Bigg\{ &\frac{D}{2}\ln\left(2\pi\sigma^2\right) + \frac{1}{2\sigma^2}\|\mathbf{y}^n\|^2 + \sum_{\phi\subseteq S}\Bigg\{\frac{Q}{2}\ln\left(2\pi\right) \\
&+ \frac{1}{2\sigma^2}\mathrm{Tr}\left(\mathbb{E}\left[\mathbf{z}_\phi^n\mathbf{z}_\phi^{n\top}\right]\mathrm{W}^\top\mathrm{W}\right) - \frac{1}{\sigma^2}\mathbb{E}\left[\mathbf{z}_\phi^n\right]^\top\mathrm{W}^\top\mathbf{y}^n \\
&+ \frac{1}{2}\ln\det\mathrm{diag}\left(\boldsymbol{\Lambda}_\phi\right) + \frac{1}{2}\mathrm{Tr}\left(\mathbb{E}\left[\mathbf{z}_\phi^n\mathbf{z}_\phi^{n\top}\right]\mathrm{diag}\left(\boldsymbol{\Lambda}_\phi\right)^{-1}\right)\Bigg\}\Bigg\}.
\end{aligned} \tag{26}$$

**Maximization Step.** Comparison of Eq. (11) and Eq. (26) shows that the maximum-likelihood estimates of $\mathrm{W} = \mathrm{U}\Gamma$ and of $\sigma^2$ are unchanged (this can be seen by substituting $\mathbf{z}$ for the sum of marginalized averages, $\sum_\phi \mathbf{z}_\phi$, so that $\mathbb{E}[\mathbf{z}] = \sum_\phi \mathbb{E}[\mathbf{z}_\phi]$ and $\mathbb{E}[\mathbf{z}\mathbf{z}^\top] = \sum_\phi \mathbb{E}[\mathbf{z}_\phi\mathbf{z}_\phi^\top]$). The maximization with respect to $\Lambda$ is more involved because we have to respect constraints from two sides. First, Eq. (21) constrains the $\mathrm{L}_1$-norm of the columns $\boldsymbol{\Lambda}_i$ of $\Lambda$. Second, since we aim for components depending only on a small subset of parameters, we have to introduce another constraint to promote sparsity of $\boldsymbol{\Lambda}_i$. Though this constraint is rather arbitrary, we found that constraining all but one entry of $\boldsymbol{\Lambda}_i$ to be zero works quite effectively, so that $\|\boldsymbol{\Lambda}_i\|_0 = 1$. Consequently, for each column $\boldsymbol{\Lambda}_i$ of $\Lambda$, the maximization of the expected likelihood, $\mathcal{L}$, Eq. (26), is given by

$$\boldsymbol{\Lambda}_i \rightarrow \arg\max_{\boldsymbol{\Lambda}_i} \mathcal{L}\left(\boldsymbol{\Lambda}_i\right) \quad \text{s.t.} \quad \|\boldsymbol{\Lambda}_i\|_1 = 1 \text{ and } \|\boldsymbol{\Lambda}_i\|_0 = 1. \tag{27}$$

Defining $B_{\phi i} = \sum_n \mathbb{E}[z_{\phi i}^n z_{\phi i}^n]$, the relevant terms in the likelihood can be written as

$$\mathcal{L}\left(\boldsymbol{\Lambda}_i\right) = -\sum_\phi \left(\ln\Lambda_{\phi i} + B_{\phi i}\Lambda_{\phi i}^{-1}\right) \tag{28}$$

$$= -\ln(1 - m\epsilon) - B_{\phi' i}(1 - m\epsilon)^{-1} - \sum_{\phi\in J}(\ln\epsilon + B_{\phi i}\epsilon^{-1}) \tag{29}$$

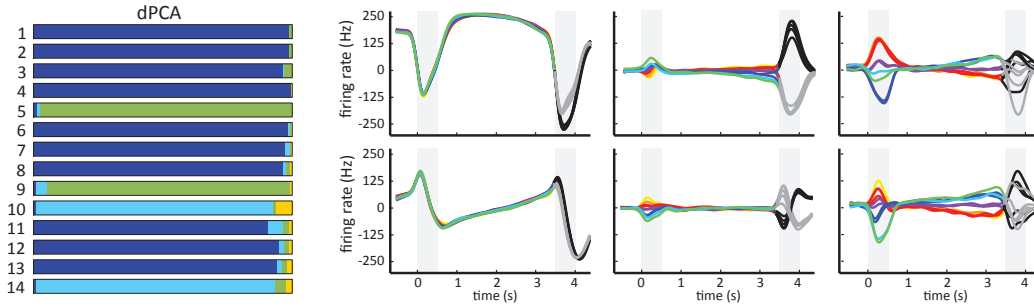

Figure 4: On the left we plot the relative variance of the fourteen highest components in dPCA conditioned on time (blue), stimulus (light blue), decision (green) and non-linear mixtures (yellow). On the right the firing rates of six dPCA components are displayed in three columns separated into components with the highest variance in time (left), in decision (middle) and in the stimulus (right).

where $\phi'$ is the index of the non-zero entry of $\mathbf{\Lambda}_i$, and $J$ is the complementing index set (of length $m = R - 1$) of all zero-entries which have been set to $\epsilon \ll 1$ for regularization purposes. Since $\epsilon$ is small, its inverse is very large. Accordingly, the likelihood is maximized for the index $\phi'$ referring to the largest entry in $B_{\phi i}$, so that

$$\Lambda_{\phi i} = \begin{cases} 1 & \text{if } \sum_n \mathbb{E}[z_{\phi i}^n z_{\phi i}^n] \geq \sum_n \mathbb{E}[z_{\psi i}^n z_{\psi i}^n] \text{ for all } \psi \neq \phi \\ 0 & \text{otherwise} \end{cases} \tag{30}$$

More generally, it is possible to substitute the sparsity constraint with $\|\mathbf{\Lambda}_i\|_0 = K$ for $K > 1$ and maximize $\mathcal{L}(\mathbf{\Lambda}_i)$ numerically. The full algorithm for dPCA is summarized in Algorithm 1.

# 6 Experimental results

The results of the dPCA algorithm applied to the electrophysiological data from the PFC are shown in Fig. 4. With 90% of the total variance in the first fourteen components, dPCA captures a comparable amount of variance as PCA (91.7%). The distribution of variances in the dPCA components is shown in Fig. 4, left. Note that, compared with the distribution in the PCA components (Fig. 1, bottom, center), the dPCA components clearly separate the different sources of variability. More specifically, the neural population is dominated by components that only depend on time (blue), yet also features separate components for the monkey's decision (green) and the perception of the stimulus (light blue). The components of dPCA, of which the six most prominent are displayed in Fig. 4, right, therefore reflect and separate the parameter dependencies of the data, even though these dependencies were completely intermingled on the single neuron level (compare Fig. 1, bottom, left).

# 7 Conclusions

Dimensionality reduction methods that take labels or parameters into account have recently found a resurgence in interest. Our study was motivated by the specific problems related to electrophysiological data sets. The main aim of our method—demixing parameter dependencies of high-dimensional data sets—may be useful in other context as well. Very similar problems arise in fMRI data, for instance, and dPCA could provide a useful alternative to other dimensionality reduction methods such as CCA, PLS, or Supervised PCA [1, 12, 5]. Furthermore, the general aim of demixing dependencies could likely be extended to other methods (such as ICA) as well. Ultimately, we see dPCA as a particular data visualization technique that will prove useful if a demixing of parameter dependencies aids in understanding data.

The source code both for Python and Matlab can be found at https://sourceforge.net/projects/dpca/.

## Footnotes

[1] The *reduced* singular value decomposition factorizes a $D \times Q$ matrix A as $\mathrm{A} = \mathrm{KDL}^*$, where K is a $D \times Q$ unitary matrix, D is a $Q \times Q$ nonnegative, real diagonal matrix, and $\mathrm{L}^*$ is a $Q \times Q$ unitary matrix.

[2]To see through this notation, notice that the $n$-th data point $\mathbf{y}_n$ or $\mathbf{y}^n$ is tagged with parameter values $\boldsymbol{\theta}^n = (\theta_{1,n}, \theta_{2,n}, \ldots)$. Any average over a subset $\psi = S \setminus \phi$ of the parameters leaves vectors $\langle\mathbf{y}\rangle_\psi$ that still depend on some remaining parameters, $\phi = \boldsymbol{\theta}_{\mathrm{rest}}$. We can therefore take their values for the $n$-th data point, $\boldsymbol{\theta}_{\mathrm{rest}}^n$, and assign the respective value of the average to the $n$-data point as well, writing $\langle\mathbf{y}\rangle_\psi^n$.

# References

[1] F. R. Bach and M. I. Jordan. A probabilistic interpretation of canonical correlation analysis. *Technical Report 688, University of California, Berkeley*, 2005.

[2] C. M. Bishop. *Pattern Recognition and Machine Learning (Information Science and Statistics)*. Springer, 2006.

[3] C. D. Brody, A. Hernández, A. Zainos, and R. Romo. Timing and neural encoding of somatosensory parametric working memory in macaque prefrontal cortex. *Cerebral Cortex*, 13(11):1196–1207, 2003.

[4] T. Hastie, R. Tibshirani, and J. Friedman. *The Elements of Statistical Learning*. Springer, 2001.

[5] A. Krishnan, L. J. Williams, A. R. McIntosh, and H. Abdi. Partial least squares (PLS) methods for neuroimaging: a tutorial and review. *NeuroImage*, 56:455–475, 2011.

[6] P.-O. Lowdin. On the non-orthogonality problem connected with the use of atomic wave functions in the theory of molecules and crystals. *The Journal of Chemical Physics*, 18(3):365, 1950.

[7] C. K. Machens. Demixing population activity in higher cortical areas. *Frontiers in computational neuroscience*, 4(October):8, 2010.

[8] C. K. Machens, R. Romo, and C. D. Brody. Functional, but not anatomical, separation of "what" and "when" in prefrontal cortex. *Journal of Neuroscience*, 30(1):350–360, 2010.

[9] R. Romo, C. D. Brody, A. Hernandez, and L. Lemus. Neuronal correlates of parametric working memory in the prefrontal cortex. *Nature*, 399(6735):470–473, 1999.

[10] S. Roweis. EM algorithms for PCA and SPCA. *Advances in neural information processing systems*, 10:626–632, 1998.

[11] M. E. Tipping and C. M. Bishop. Probabilistic principal component analysis. *Journal of the Royal Statistical Society - Series B: Statistical Methodology*, 61(3):611–622, 1999.

[12] S. Yu, K. Yu, V. Tresp, H. P. Kriegel, and M. Wu. Supervised probabilistic principal component analysis. *Proceedings of 12th ACM SIGKDD International Conf. on KDD*, 10, 2006.

